# Inferring State Sequences for Non-linear Systems with Embedded Hidden Markov Models

**Radford M. Neal, Matthew J. Beal, and Sam T. Roweis**
Department of Computer Science
University of Toronto
Toronto, Ontario, Canada M5S 3G3
{radford,beal,roweis}@cs.utoronto.ca

## Abstract

We describe a Markov chain method for sampling from the distribution of the hidden state sequence in a non-linear dynamical system, given a sequence of observations. This method updates all states in the sequence simultaneously using an embedded Hidden Markov Model (HMM). An update begins with the creation of "pools" of candidate states at each time. We then define an embedded HMM whose states are indexes within these pools. Using a forward-backward dynamic programming algorithm, we can efficiently choose a state sequence with the appropriate probabilities from the exponentially large number of state sequences that pass through states in these pools. We illustrate the method in a simple one-dimensional example, and in an example showing how an embedded HMM can be used to in effect discretize the state space without any discretization error. We also compare the embedded HMM to a particle smoother on a more substantial problem of inferring human motion from 2D traces of markers.

## 1  Introduction

Consider a dynamical model in which a sequence of hidden states, $\boldsymbol{x} = (x_0, \ldots, x_{n-1})$, is generated according to some stochastic transition model. We observe $\boldsymbol{y} = (y_0, \ldots, y_{n-1})$, with each $y_t$ being generated from the corresponding $x_t$ according to some stochastic observation process. Both the $x_t$ and the $y_t$ could be multidimensional. We wish to randomly sample hidden state sequences from the conditional distribution for the state sequence given the observations, which we can then use to make Monte Carlo inferences about this posterior distribution for the state sequence. We suppose in this paper that we know the dynamics of hidden states and the observation process, but if these aspects of the model are unknown, the method we describe will be useful as part of a maximum likelihood learning algorithm such as EM, or a Bayesian learning algorithm using Markov chain Monte Carlo.

If the state space is finite, of size $K$, so that this is a Hidden Markov Model (HMM), a hidden state sequence can be sampled by a forward-backwards dynamic programming algorithm in time proportional to $nK^2$ (see [5] for a review of this and related algorithms). If the state space is $\Re^p$ and the dynamics and observation process are linear, with Gaussian noise, an analogous adaptation of the Kalman filter can be used. For more general models,

or for finite state space models in which $K$ is large, one might use Markov chain sampling (see [3] for a review). For instance, one could perform Gibbs sampling or Metropolis updates for each $x_t$ in turn. Such simple Markov chain updates may be very slow to converge, however, if the states at nearby times are highly dependent. A popular recent approach is to use a particle smoother, such as the one described by Doucet, Godsill, and West [2], but this approach can fail when the set of particles doesn't adequately cover the space, or when particles are eliminated prematurely.

In this paper, we present a Markov chain sampling method for a model with an arbitrary state space, $\mathcal{X}$, in which efficient sampling is facilitated by using updates that are based on temporarily embedding an HMM whose finite state space is a subset of $\mathcal{X}$, and then applying the efficient HMM sampling procedure. We illustrate the method on a simple one-dimensional example. We also show how it can be used to in effect discretize the state space without producing any discretization error. Finally, we demonstrate the embedded HMM on a problem of tracking human motion in 3D based on the 2D projections of marker positions, and compare it with a particle smoother.

## 2   The Embedded HMM Algorithm

In our description of the algorithm, model probabilities will be denoted by $P$, which will denote probabilities or probability densities without distinction, as appropriate for the state space, $\mathcal{X}$, and observation space, $\mathcal{Y}$. The model's initial state distribution is given by $P(x_0)$, transition probabilities are given by $P(x_t \mid x_{t-1})$, and observation probabilities are given by $P(y_t \mid x_t)$. Our goal is to sample from the conditional distribution $P(x_0, \ldots, x_{n-1} \mid y_0, \ldots, y_{n-1})$, which we will abbreviate to $\pi(x_0, \ldots, x_{n-1})$, or $\pi(\boldsymbol{x})$.

To accomplish this, we will simulate a Markov chain whose state space is $\mathcal{X}^n$ — i.e., a state of this chain is an entire sequence of hidden states. We will arrange for the equilibrium distribution of this Markov chain to be $\pi(x_0, \ldots, x_{n-1})$, so that simulating the chain for a suitably long time will produce a state sequence from the desired distribution. The state at iteration $i$ of this chain will be written as $\boldsymbol{x}^{(i)} = (x_0^{(i)}, \ldots, x_{n-1}^{(i)})$. The transition probabilities for this Markov chain will be denoted using $Q$. In particular, we will use some initial distribution for the state of the chain, $Q(\boldsymbol{x}^{(0)})$, and will simulate the chain according to the transition probabilities $Q(\boldsymbol{x}^{(i)} \mid \boldsymbol{x}^{(i-1)})$. For validity of the sampling method, we need these transitions to leave $\pi$ invariant:

$$\pi(\boldsymbol{x}') \;\; = \;\; \sum_{\boldsymbol{x} \, \in \, \mathcal{X}^n} \pi(\boldsymbol{x}) Q(\boldsymbol{x}' \mid \boldsymbol{x}), \quad \text{for all } \boldsymbol{x}' \text{ in } \mathcal{X}^n \tag{1}$$

(If $\mathcal{X}$ is continuous, the sum is replaced by an integral.) This is implied by the detailed balance condition:

$$\pi(\boldsymbol{x}) Q(\boldsymbol{x}' \mid \boldsymbol{x}) \;\; = \;\; \pi(\boldsymbol{x}') Q(\boldsymbol{x} \mid \boldsymbol{x}'), \quad \text{for all } \boldsymbol{x} \text{ and } \boldsymbol{x}' \text{ in } \mathcal{X}^n \tag{2}$$

The transition $Q(\boldsymbol{x}^{(i)} \mid \boldsymbol{x}^{(i-1)})$ is defined in terms of "pools" of states for each time. The current state at time $t$ is always part of the pool for time $t$. Other states in the pool are produced using a pool distribution, $\rho_t$, which is designed so that points drawn from $\rho_t$ are plausible alternatives to the current state at time $t$. The simplest way to generate these additional pool states is to draw points independently from $\rho_t$. This may not be feasible, however, or may not be desirable, in which case we can instead simulate an "inner" Markov chain defined by transition probabilities written as $R_t(\cdot \mid \cdot)$, which leave the pool distribution, $\rho_t$, invariant. The transitions for the reversal of this chain with respect to $\rho_t$ will be denoted by $\tilde{R}_t(\cdot \mid \cdot)$, and are defined so as to satisfy the following condition:

$$\rho_t(x_t) R_t(x_t' \mid x_t) \;\; = \;\; \rho_t(x_t') \tilde{R}_t(x_t \mid x_t'), \quad \text{for all } x_t \text{ and } x_t' \text{ in } \mathcal{X} \tag{3}$$

If the transitions $R_t$ satisfy detailed balance with respect to $\rho_t$, $\tilde{R}_t$ will be the same as $R_t$. To generate pool states by drawing from $\rho_t$ independently, we can let $R_t(x'|x) = \tilde{R}_t(x'|x) = \rho_t(x')$. For the proof of correctness below, we must not choose $\rho_t$ or $R_t$ based on the current state, $\boldsymbol{x}^{(i)}$, but we may choose them based on the observations, $\boldsymbol{y}$.

To perform a transition $Q$ to a new state sequence, we begin by at each time, $t$, producing a pool of $K$ states, $\mathcal{C}_t$. One of the states in $\mathcal{C}_t$ is the current state, $x_t^{(i-1)}$; the others are produced using $R_t$ and $\tilde{R}_t$. The new state sequence, $\boldsymbol{x}^{(i)}$, is then randomly selected from among all sequences whose states at each time $t$ are in $\mathcal{C}_t$, using a form of the forward-backward procedure.

In detail, the pool of candidate states for time $t$ is found as follows:

1) Pick an integer $J_t$ uniformly from $\{0, \ldots, K-1\}$.

2) Let $x_t^{[0]} = x_t^{(i-1)}$. (So the current state is always in the pool.)

3) For $j$ from 1 to $J_t$, randomly pick $x_t^{[j]}$ according to the transition probabilities $R_t(x_t^{[j]} \mid x_t^{[j-1]})$.

4) For $j$ from $-1$ down to $-K + J_t + 1$, randomly pick $x_t^{[j]}$ according to the reversed transition probabilities, $\tilde{R}_t(x_t^{[j]} \mid x_t^{[j+1]})$.

5) Let $\mathcal{C}_t$ be the pool consisting of $x_t^{[j]}$, for $j \in \{-K+J_t+1, \ldots, 0, \ldots, J_t\}$. If some of the $x_t^{[j]}$ are the same, they will be present in the pool more than once.

Once the pools of candidate states have been found, a new state sequence, $\boldsymbol{x}^{(i)}$, is picked from among all sequences, $\boldsymbol{x}$, for which every $x_t$ is in $\mathcal{C}_t$. The probability of picking $\boldsymbol{x}^{(i)} = \boldsymbol{x}$ is proportional to $\pi(\boldsymbol{x})/\prod_{t=0}^{n-1} \rho_t(x_t)$, which is proportional to

$$\frac{P(x_0) \prod_{t=1}^{n-1} P(x_t \mid x_{t-1}) \prod_{t=0}^{n-1} P(y_t \mid x_t)}{\prod_{t=0}^{n-1} \rho_t(x_t)} \tag{4}$$

The division by $\prod_{t=0}^{n-1} \rho_t(x_t)$ is needed to compensate for the pool states having been drawn from the $\rho_t$ distributions. If duplicate states occur in some of the pools, they are treated as if they were distinct when picking a sequence in this way. In effect, we pick indexes of states in these pools, with probabilities as above, rather than states themselves.

The distribution of these sequences of indexes can be regarded as the posterior distribution for a hidden Markov model, with the transition probability from state $j$ at time $t-1$ to state $k$ at time $t$ being proportional to $P(x_t^{[k]} \mid x_{t-1}^{[j]})$, and the probabilities of the hypothetical observed symbols being proportional to $P(y_t \mid x_t^{[k]})/\rho_t(x_t^{[k]})$. Crucially, using the forward-backward technique, it is possible to randomly pick a new state sequence from this distribution in time growing linearly with $n$, even though the number of possible sequences grows as $K^n$. After the above procedure has been used to produce the pool states, $x_t^{[j]}$ for $t = 0$ to $n-1$ and $j = -K + J_t + 1$ to $J_t$, this algorithm operates as follows (see [5]):

1) For $t = 0$ to $n-1$ and for $j = -K+J_t+1$ to $J_t$, let $u_{t,j} = P(y_t \mid x_t^{[j]})/\rho_t(x_t^{[j]})$.

2) For $j = -K+J_0+1$ to $J_0$, let $w_{0,j} = u_{0,j} P(X_0 = x_0^{[j]})$.

3) For $t = 1$ to $n-1$ and for $j = -K+J_t + 1$ to $J_t$, let

$$w_{t,j} = u_{t,j} \sum_k w_{t-1,k} P(x_t^{[j]} \mid x_{t-1}^{[k]})$$

4) Randomly pick $s_{n-1}$ from $\{-K+J_{n-1}+1, \ldots, J_{n-1}\}$, picking the value $j$ with probability proportional to $w_{n-1,j}$.

5) For $t = n-1$ down to 1, randomly pick $s_{t-1}$ from $\{-K+J_{t-1}+1, \ldots, J_{t-1}\}$, picking the value $j$ with probability proportional to $w_{t-1,j} \, P(x_t^{[s_t]} \,|\, x_{t-1}^{[j]})$.

Note that when implementing this algorithm, one must take some measure to avoid floating-point underflow, such as representing the $w_{t,j}$ by their logarithms.

Finally, the embedded HMM transition is completed by letting the new state sequence, $\boldsymbol{x}^{(i)}$, be equal to $(x_0^{[s_0]}, x_1^{[s_1]}, \ldots, x_{n-1}^{[s_{n-1}]})$

## 3 Proof of Correctness

To show that a Markov chain with these transitions will converge to $\pi$, we need to show that it leaves $\pi$ invariant, and that the chain is ergodic. Ergodicity need not always hold, and proving that it does hold may require considering the particulars of the model. However, it is easy to see that the chain will be ergodic if all possible state sequences have non-zero probability density under $\pi$, the pool distributions, $\rho_t$, have non-zero density everywhere, and the transitions $R_t$ are ergodic. This probably covers most problems that arise in practice.

To show that the transitions $Q(\cdot \,|\, \cdot)$ leave $\pi$ invariant, it suffices to show that they satisfy detailed balance with respect to $\pi$. This will follow from the stronger condition that the probability of moving from $\boldsymbol{x}$ to $\boldsymbol{x}'$ (starting from a state picked from $\pi$) with given values for the $J_t$ and given pools of candidate states, $\mathcal{C}_t$, is the same as the corresponding probability of moving from $\boldsymbol{x}'$ to $\boldsymbol{x}$ with the same pools of candidate states and with values $J_t'$ defined by $J_t' = J_t - h_t$, where $h_t$ is the index (from $-K + J_t + 1$ to $J_t$) of $x_t'$ in the candidate pool.

The probability of such a move from $\boldsymbol{x}$ to $\boldsymbol{x}'$ is the product of several factors. First, there is the probability of starting from $\boldsymbol{x}$ under $\pi$, which is $\pi(\boldsymbol{x})$. Then, for each time $t$, there is the probability of picking $J_t$, which is $1/K$, and of then producing the states in the candidate pool using the transitions $R_t$ and $\tilde{R}_t$, which is

$$\prod_{j=1}^{J_t} R_t(x_t^{[j]} \,|\, x_t^{[j-1]}) \ \times \ \prod_{j=-K+J_t+1}^{-1} \tilde{R}_t(x_t^{[j]} \,|\, x_t^{[j+1]})$$

$$= \ \prod_{j=0}^{J_t-1} R_t(x_t^{[j+1]} \,|\, x_t^{[j]}) \ \times \ \prod_{j=-K+J_t+1}^{-1} R_t(x_t^{[j+1]} \,|\, x_t^{[j]}) \, \frac{\rho_t(x_t^{[j]})}{\rho_t(x_t^{[j+1]})} \tag{5}$$

$$= \ \frac{\rho_t(x_t^{[-K+J_t+1]})}{\rho_t(x_t^{[0]})} \ \prod_{j=-K+J_t+1}^{J_t-1} R_t(x_t^{[j+1]} \,|\, x_t^{[j]}) \tag{6}$$

Finally, there is the probability of picking $\boldsymbol{x}'$ from among all the sequences with states from the pools, $\mathcal{C}_t$, which is proportional to $\pi(\boldsymbol{x}') / \prod \rho_t(x_t')$. The product of all these factors is

$$\pi(\boldsymbol{x}) \ \times \ \frac{1}{K^n} \ \times \ \prod_{t=0}^{n-1} \left[ \frac{\rho_t(x_t^{[-K+J_t+1]})}{\rho_t(x_t^{[0]})} \prod_{j=-K+J_t+1}^{J_t-1} R_t(x_t^{[j+1]} \,|\, x_t^{[j]}) \right] \ \times \ \frac{\pi(\boldsymbol{x}')}{\prod_{t=0}^{n-1} \rho_t(x_t')}$$

$$= \ \frac{1}{K^n} \, \frac{\pi(\boldsymbol{x})\pi(\boldsymbol{x}')}{\prod_{t=0}^{n-1} \rho(x_t)\rho(x_t')} \ \prod_{t=0}^{n-1} \left[ \rho_t(x_t^{[-K+J_t+1]}) \prod_{j=-K+J_t+1}^{J_t-1} R_t(x_t^{[j+1]} \,|\, x_t^{[j]}) \right] \tag{7}$$

We can now see that the corresponding expression for a move from $\boldsymbol{x}'$ to $\boldsymbol{x}$ is identical, apart from a relabelling of candidate state $x_t^{[j]}$ as $x_t^{[j-h_t]}$.

## 4   A simple demonstration

The following simple example illustrates the operation of the embedded HMM. The state space $\mathcal{X}$ and the observation space, $\mathcal{Y}$, are both $\Re$, and each observation is simply the state plus Gaussian noise of standard deviation $\sigma$ — i.e., $P(y_t \mid x_t) = N(y_t \mid x_t, \sigma^2)$. The state transitions are defined by $P(x_t \mid x_{t-1}) = N(x_t \mid \tanh(\eta x_{t-1}), \tau^2)$, for some constant expansion factor $\eta$ and transition noise standard deviation $\tau$.

Figure 1 shows a hidden state sequence, $x_0, \ldots, x_{n-1}$, and observation sequence, $y_0, \ldots, y_{n-1}$, generated by this model using $\sigma = 2.5$, $\eta = 2.5$, and $\tau = 0.4$, with $n = 1000$. The state sequence stays in the vicinity of $+1$ or $-1$ for long periods, with rare switches between these regions. Because of the large observation noise, there is considerable uncertainty regarding the state sequence given the observation sequence, with the posterior distribution assigning fairly high probability to sequences that contain short-term switches between the $+1$ and $-1$ regions that are not present in the actual state sequence, or that lack some of the short-term switches that are actually present.

We sampled from this distribution over state sequences using an embedded HMM in which the pool distributions, $\rho_t$, were normal with mean zero and standard deviation one, and the pool transitions simply sampled independently from this distribution (ignoring the current pool state). Figure 2 shows that after only two updates using pools of ten states, embedded HMM sampling produces a state sequence with roughly the correct characteristics. Figure 3 demonstrates how a single embedded HMM update can make a large change to the state sequence. It shows a portion of the state sequence after 99 updates, the pools of states produced for the next update, and the state sequence found by the embedded HMM using these pools. A large change is made to the state sequence in the region from time 840 to 870, with states in this region switching from the vicinity of $-1$ to the vicinity of $+1$.

This example is explored in more detail in [4], where it is shown that the embedded HMM is superior to simple Metropolis methods that update one hidden state at a time.

## 5   Discretization without discretization error

A simple way to handle a model with a continuous state space is to discretize the space by laying down a regular grid, after transforming to make the space bounded if necessary. An HMM with grid points as states can then be built that approximates the original model. Inference using this HMM is only approximate, however, due to the discretization error involved in replacing the continuous space by a grid of points.

The embedded HMM can use a similar grid as a deterministic method of creating pools of states, aligning the grid so that the current state lies on a grid point. This is a special case of the general procedure for creating pools, in which $\rho_t$ is uniform, $R_t$ moves to the next grid point and $\tilde{R}_t$ moves to the previous grid point, with both wrapping around when the first or last grid point is reached. If the number of pool states is set equal to the number of points in a grid, every pool will consist of a complete grid aligned to include the current state.

On their own, such embedded HMM updates will never change the alignments of the grids. However, we can alternately apply such an embedded HMM update and some other MCMC update (eg, Metropolis) which is capable of making small changes to the state. These small changes will change the alignment of the new grids, since each grid is aligned to include the current state. The combined chain will be ergodic, and sample (asymptotically) from the correct distribution. This method uses a grid, but nevertheless has no discretization error.

We have tried this method on the example described above, laying the grid over the transformed state $\tanh(x_t)$, with suitably transformed transition densities. With $K = 10$, the grid method samples more efficiently than when using $N(0, 1)$ pool distributions, as above.

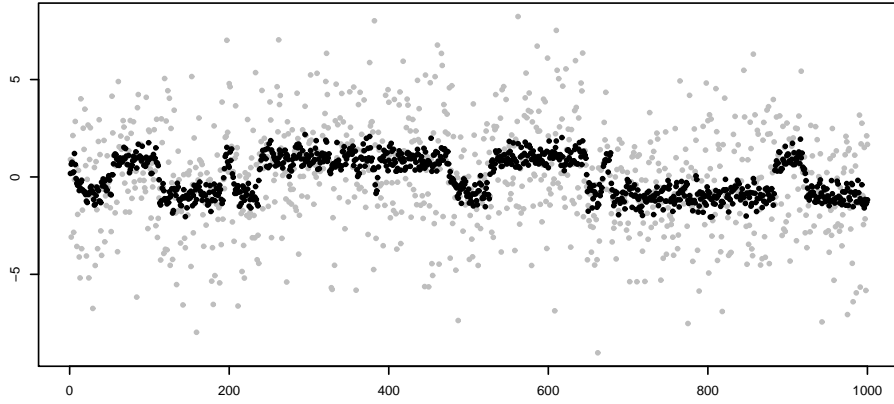

Figure 1: A state sequence (black dots) and observation sequence (gray dots) of length 1000 produced by the model with $\sigma = 2.5$, $\eta = 2.5$, and $\tau = 0.4$.

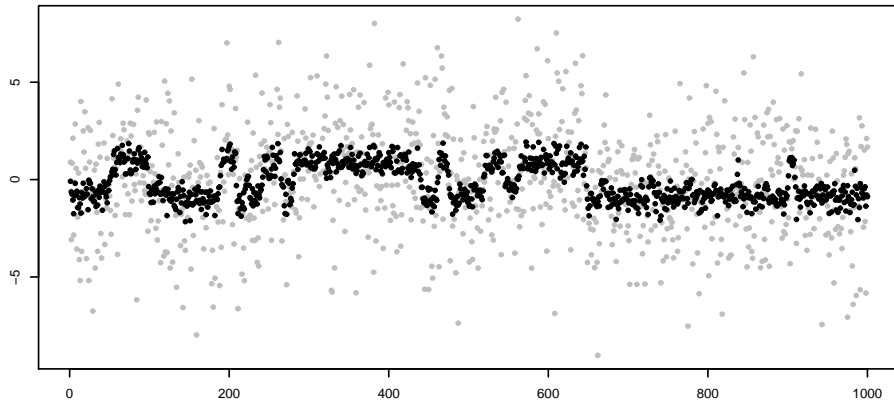

Figure 2: The state sequence (black dots) produced after two embedded HMM updates, starting with the states set equal to the data points (gray dots), as in the figure above.

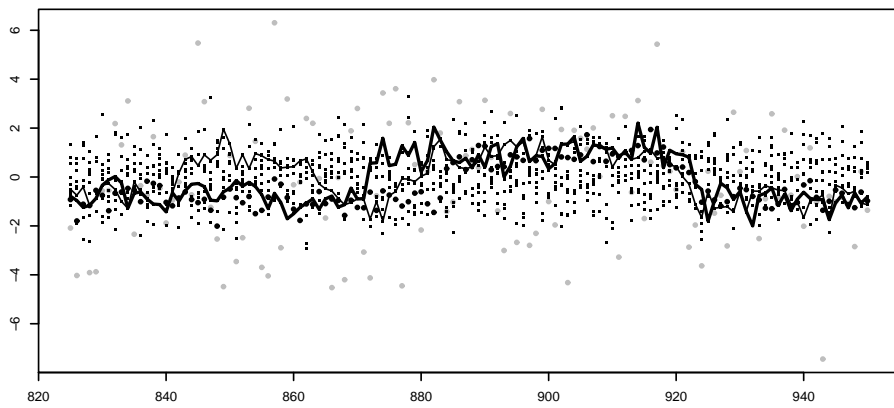

Figure 3: Closeup of an embedded HMM update. The true state sequence is shown by black dots and the observation sequence by gray dots. The current state sequence is shown by the dark line. The pools of ten states at each time used for the update are shown as small dots, and the new state sequence picked by the embedded HMM by the light line.

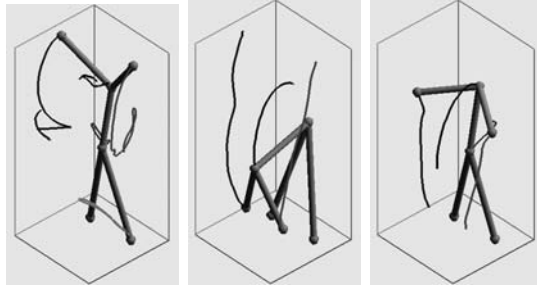

Figure 4: The four-second motion sequence used for the experiment, shown in three snapshots with streamers showing earlier motion. The left plot shows frames 1-59, the middle plot frames 59-91, and the right plot frames 91-121. There were 30 frames per second. The orthographic projection in these plots is the one seen by the model. (These plots were produced using Hertzmann and Brand's *mosey* program.)

## 6    Tracking human motion

We have applied the embedded HMM to the more challenging problem of tracking 3D human motion from 2D observations of markers attached to certain body points. We constructed this example using real motion-capture data, consisting of the 3D positions at each time frame of a set of identified markers. We chose one subject, and selected six markers (on left and right feet, left and right hands, lower back, and neck). These markers were projected to a 2D viewing plane, with the viewing direction being known to the model. Figure 4 shows the four-second sequence used for the experiment.[1]

Our goal was to recover the 3D motion of the six markers, by using the embedded HMM to generate samples from the posterior distribution over 3D positions at each time (the hidden states of the model), given the 2D observations. To do this, we need some model of human dynamics. As a crude approximation, we used Langevin dynamics with respect to a simple hand-designed energy function that penalizes unrealistic body positions. In Langevin dynamics, a gradient descent step in the energy is followed by the addition of Gaussian noise, with variance related to the step size. The equilibrium distribution for this dynamics is the Boltzmann distribution for the energy function. The energy function we used contains terms pertaining to the pairwise distances between the six markers and to the heights of the markers above the plane of the floor, as well as a term that penalizes bending the torso far backwards while the legs are vertical. We chose the step size for the Langevin dynamics to roughly match the characteristics of the actual data.

The embedded HMM was initialized by setting the state at all times to a single frame of the subject in a typical stance, taken from a different trial. As the pool distribution at time $t$, we used the posterior distribution when using the Boltzmann distribution for the energy as the prior and the single observation at time $t$. The pool transitions used were Langevin updates with respect to this pool distribution.

For comparison, we also tried solving this problem with the particle smoother of [2], in which a particle filter is applied to the data in time order, after which a state sequence is selected at random in a backwards pass. We used a stratified resampling method to reduce variance. The initial particle set was created by drawing frames randomly from sequences other than the sequence being tested, and translating the markers in each frame so that their centre of mass was at the same point as the centre of mass in the test sequence.

Both programs were implemented in MATLAB. The particle smoother was run with 5000 particles, taking 7 hours of compute time. The resulting sampled trajectories roughly fit the 2D observations, but were rather unrealistic — for instance, the subject's feet often floated above the floor. We ran the embedded HMM using five pool states for 300 iterations, taking 1.7 hours of compute time. The resulting sampled trajectories were more realistic

than those produced by the particle smoother, and were quantitatively better with respect to likelihood and dynamical transition probabilities. However, the distribution of trajectories found did not overlap the true trajectory. The embedded HMM updates appeared to be sampling from the correct posterior distribution, but moving rather slowly among those trajectories that are plausible given the observations.

## 7   Conclusions

We have shown that the embedded HMM can work very well for a non-linear model with a low-dimensional state. For the higher-dimensional motion tracking example, the embedded HMM has some difficulties exploring the full posterior distribution, due, we think, to the difficulty of creating pool distributions with a dense enough sampling of states to allow linking of new states at adjacent times. However, the particle smoother was even more severely affected by the high dimensionality of this problem. The embedded HMM therefore appears to be a promising alternative to particle smoothers in such contexts.

The idea behind the embedded HMM should also be applicable to more general tree-structured graphical models. A pool of values would be created for each variable in the tree (which would include the current value for the variable). The fast sampling algorithm possible for such an "embedded tree" (a generalization of the sampling algorithm used for the embedded HMM) would then be used to sample a new set of values for all variables, choosing from all combinations of values from the pools.

Finally, while much of the elaboration in this paper is designed to create a Markov chain whose equilibrium distribution is exactly the correct posterior, $\pi(\boldsymbol{x})$, the embedded HMM idea can be also used as a simple search technique, to find a state sequence, $\boldsymbol{x}$, which maximizes $\pi(\boldsymbol{x})$. For this application, any method is acceptable for proposing pool states (though some proposals will be more useful than others), and the selection of a new state sequence from the resulting embedded HMM is done using a Viterbi-style dynamic programming algorithm that selects the trajectory through pool states that maximizes $\pi(\boldsymbol{x})$. If the current state at each time is always included in the pool, this Viterbi procedure will always either find a new $\boldsymbol{x}$ that increases $\pi(\boldsymbol{x})$, or return the current $\boldsymbol{x}$ again. This embedded HMM optimizer has been successfully used to infer segment boundaries in a segmental model for voicing detection and pitch tracking in speech signals [1], as well as in other applications such as robot localization from sensor logs.

**Acknowledgments**. This research was supported by grants from the Natural Sciences and Engineering Research Council of Canada, and by an Ontario Premier's Research Excellence Award. Computing resources were provided by a CFI grant to Geoffrey Hinton.

## Footnotes

[1]Data from the graphics lab of Jessica Hodgins, at `http://mocap.cs.cmu.edu`. We chose markers 167, 72, 62, 63, 31, 38, downsampled to 30 frames per second. The experiments reported here use frames 400-520 of trial 20 for subject 14. The elevation of the view direction was 45 degrees, and the azimuth was 45 degrees away from a front view of the person in the first frame.

## References

[1]   Achan, K., Roweis, S. T., and Frey, B. J. (2004) "A Segmental HMM for Speech Waveforms", Technical Report UTML-TR-2004-001, University of Toronto, January 2004.

[2]   Doucet, A., Godsill, S. J., and West, M. (2000) "Monte Carlo filtering and smoothing with application to time-varying spectral estimation" *Proc. IEEE International Conference on Acoustics, Speech and Signal Processing, 2000*, volume II, pages 701-704.

[3]   Neal, R. M. (1993) *Probabilistic Inference Using Markov Chain Monte Carlo Methods*, Technical Report CRG-TR-93-1, Dept. of Computer Science, University of Toronto, 144 pages. Available from http://www.cs.utoronto.ca/~radford.

[4]   Neal, R. M. (2003) "Markov chain sampling for non-linear state space models using embedded hidden Markov models", Technical Report No. 0304, Dept. of Statistics, University of Toronto, 9 pages. Available from http://www.cs.utoronto.ca/~radford.

[5]   Scott, S. L. (2002) "Bayesian methods for hidden Markov models: Recursive computing in the 21st century", *Journal of the American Statistical Association*, vol. 97, pp. 337–351.